# Bayesian averaging is well-temperated

**Lars Kai Hansen**
Department of Mathematical Modelling
Technical University of Denmark B321
DK-2800 Lyngby, Denmark
*lkhansen@imm.dtu.dk*

## Abstract

Bayesian predictions are stochastic just like predictions of any other inference scheme that generalize from a finite sample. While a simple variational argument shows that Bayes averaging is generalization optimal given that the prior matches the teacher parameter distribution the situation is less clear if the teacher distribution is unknown. I define a class of averaging procedures, the temperated likelihoods, including both Bayes averaging with a uniform prior and maximum likelihood estimation as special cases. I show that Bayes is generalization optimal in this family for any teacher distribution for two learning problems that are analytically tractable: learning the mean of a Gaussian and asymptotics of smooth learners.

## 1 Introduction

Learning is the stochastic process of generalizing from a random finite sample of data. Often a learning problem has natural quantitative measure of generalization. If a loss function is defined the natural measure is the *generalization error*, i.e., the *expected loss* on a random sample independent of the training set. Generalizability is a key topic of learning theory and much progress has been reported. Analytic results for a broad class of machines can be found in the litterature [8, 12, 9, 10] describing the asymptotic generalization ability of supervised algorithms that are continuously parameterized. Asymptotic bounds on generalization for general machines have been advocated by Vapnik [11]. Generalization results valid for finite training sets can only be obtained for specific learning machines, see e.g. [5]. A very rich framework for analysis of generalization for Bayesian averaging and other schemes is defined in [6].

Averaging has become popular as a tool for improving generalizability of learning machines. In the context of (time series) forecasting averaging has been investigated intensely for decades [3]. Neural network ensembles were shown to improve generalization by simple voting in [4] and later work has generalized these results to other types of averaging. Boosting, Bagging, Stacking, and Arcing are recent examples of averaging procedures based on data resampling that have shown useful see [2] for a recent review with references. However, Bayesian averaging in particular is attaining a kind of cult status. Bayesian averaging is indeed provably optimal in a

number various ways (admissibility, the likelihood principle etc) [1]. While it follows by construction that Bayes is generalization optimal if given the correct prior information, i.e., the teacher parameter distribution, the situation is less clear if the teacher distribution is unknown. Hence, the pragmatic Bayesians downplay the role of the prior. Instead the averaging aspect is emphasized and "vague" priors are invoked. It is important to note that whatever prior is used Bayesian predictions are stochastic just like predictions of any other inference scheme that generalize from a finite sample.

In this contribution I analyse two scenarios where averaging can improve generalizability and I show that the vague Bayes average is in fact optimal among the averaging schemes investigated. Averaging is shown to reduce variance at the cost of introducing bias, and Bayes happens to implement the optimal bias-variance trade-off.

## 2  Bayes and generalization

Consider a model that is smoothly parametrized and whose predictions can be described in terms of a density function[1]. Predictions in the model are based on a given training set: a finite sample $D = \{x_\alpha\}_{\alpha=1}^N$ of the stochastic vector $x$ whose density – the teacher – is denoted $p(x|\theta_0)$. In other words the true density is assumed to be defined by a fixed, but unknown, teacher parameter vector $\theta_0$. The model, denoted $H$, involves the parameter vector $\theta$ and the predictive density is given by

$$p(x|D, H) = \int p(x|\theta, H)p(\theta|D, H)d\theta \qquad (1)$$

$p(\theta|D, H)$ is the parameter distribution produced in training process. In a maximum likelihood scenario this distribution is a delta function centered on the most likely parameters under the model for the given data set. In ensemble averaging approaches, like boosting bagging or stacking, the distribution is obtained by training on resampled traning sets. In a Bayesian scenario, the parameter distribution is the posterior distribution,

$$p(\theta|D, H) = \frac{p(D|\theta, H)p(\theta|H)}{\int p(D|\theta', H)p(\theta'|H)d\theta'} \qquad (2)$$

where $p(\theta|H)$ is the prior distribution (probability density of parameters if D is empty). In the sequel we will only consider one model hence we suppress the model conditioning label $H$.

The generalization error is the average negative log density (also known as simply the "log loss" – in some applied statistics works known as the "deviance")

$$\Gamma(D|\theta_0) = \int -\log p(x|D)p(x|\theta_0)dx, \qquad (3)$$

The expected value of the generalization error for training sets produced by the given teacher is given by

$$\Gamma(\theta_0) = \int \int -\log p(x|D)p(x|\theta_0)dx p(D|\theta_0)dD. \qquad (4)$$

Playing the game of "guessing a probability distribution" [6] we not only face a random training set, we also face a teacher drawn from the teacher distribution $p(\theta_0)$. The teacher averaged generalization must then be defined as

$$\Gamma = \int \Gamma(\theta_0)p(\theta_0)d\theta_0. \tag{5}$$

This is the typical generalization error for a random training set from the randomly chosen teacher – produced by the model $H$. The generalization error is minimized by Bayes averaging if the teacher distribution is used as prior. To see this, form the Lagrangian functional

$$\mathcal{L}[q(x|D)] = \int \int \int -\log q(x|D)p(x|\theta_0)dx p(D|\theta_0)dD p(\theta_0)d\theta_0 + \lambda \int q(x|D)dx \tag{6}$$

defined on positive functions $q(x|D)$. The second term is used to ensure that $q(x|D)$ is a normalized density in $x$. Now compute the variational derivative to obtain

$$\frac{\delta\mathcal{L}}{\delta q(x|D)} = -\frac{1}{q(x|D)}\int p(x|\theta_0)p(D|\theta_0)p(\theta_0)d\theta_0 + \lambda. \tag{7}$$

Equating this derivative to zero we recover the predictive distribution of Bayesian averaging,

$$q(x|D) = \int p(x|\theta)\frac{p(D|\theta)p(\theta)}{\int p(D|\theta')p(\theta')d\theta'}d\theta, \tag{8}$$

where we used that $\lambda = \int p(D|\theta)p(\theta)d\theta$ is the appropriate normalization constant. It is easily verified that this is indeed the global minimum of the averaged generalization error. We also note that if the Bayes average is performed with another prior than the teacher distribution $p(\theta_0)$, we can expect a higher generalization error. The important question from a Bayesian point of view is then: Are there cases where averaging with generic priors (e.g. vague or uniform priors) can be shown to be optimal?

## 3 Temperated likelihoods

To come closer to a quantative statement about when and why vague Bayes is the better procedure we will analyse two problems for which some analytical progress is possible. We will consider a one-parameter family of learning procedures including both a Bayes and the maximum likelihood procedure,

$$p(\theta|\beta, D, H) = \frac{p^\beta(D|\theta)}{\int p^\beta(D|\theta')d\theta'}, \tag{9}$$

where $\beta$ is a positive parameter (plying the role of an inverse temperature). The family of procedures are all averaging procedures, and $\beta$ controls the width of the average. Vague Bayes (here used synonymously with Bayes with a uniform prior) is recoved for $\beta = 1$, while the maximum posterior procedure is obtained by cooling to zero width $\beta \to \infty$.

In this context the generalization design question can be frased as follows: *is there an optimal temperature in the family of the temperated likelihoods?*

### 3.1 Example: 1D normal variates

Let the teacher distribution be given by

$$p(x|\theta_0) = \frac{1}{\sqrt{2\pi\sigma^2}}\exp\left(-\frac{1}{2\sigma^2}(x - \theta_0)^2\right) \tag{10}$$

The model density is of the same form with $\theta$ unknown and $\sigma^2$ assumed to be known. For $N$ examples the posterior (with a uniform prior) is,

$$p(\theta|D) = \sqrt{\frac{N}{2\pi\sigma^2}} \exp\left(-\frac{N}{2\sigma^2}(\overline{x} - \theta)^2\right), \qquad (11)$$

with $\overline{x} = 1/N \sum_\alpha x_\alpha$. The temperated likelihood is obtained by raising to the $\beta$'th power and normalizing,

$$p(\theta|D, \beta) = \sqrt{\frac{\beta N}{2\pi\sigma^2}} \exp\left(-\frac{\beta N}{2\sigma^2}(\overline{x} - \theta)^2\right). \qquad (12)$$

The predictive distribution is found by integrating w.r.t. $\theta$,

$$p(x|D, \beta) = \int p(x|\theta)p(\theta|D, \beta)d\theta = \frac{1}{\sqrt{2\pi\sigma_\beta^2}} \exp\left(-\frac{1}{2\sigma_\beta^2}(\overline{x} - x)^2\right), \qquad (13)$$

with $\sigma_\beta^2 = \sigma^2(1 + 1/\beta N)$. We note that this distribution is wider for all the averaging procedures than it is for maximum likelihood ($\beta \to \infty$), i.e., less variant. For very small $\beta$ the predictive distribution is almost independent of the data set, hence highly biased.

It is straightforward to compute the generalization error of the predictive distribution for general $\beta$. First we compute the generalization error for the specific training set $D$,

$$\Gamma(D, \beta, \theta_0) = \int -\log p(x|D, \beta)p(x|\theta_0)dx = \log\sqrt{2\pi\sigma_\beta^2} + \frac{1}{2\sigma_\beta^2}\left((\overline{x} - \theta_0)^2 + \sigma^2\right), \qquad (14)$$

The average generalization error is then found by averaging w.r.t the sampling distribution using $\overline{x} \sim \mathcal{N}(\theta_0, \sigma^2/N)$.,

$$\Gamma(\beta) = \int \Gamma(D, \beta)dD p(D|\theta_0) = \log\sqrt{2\pi\sigma_\beta^2} + \frac{\sigma^2}{2\sigma_\beta^2}\left(\frac{1}{N} + 1\right), \qquad (15)$$

We first note that the generalization error is independent of the teacher $\theta_0$ parameter, this happened because $\theta$ is a "location" parameter. The $\beta$-dependency of the averaged generalization error is depicted in Figure 1. Solving $\partial\Gamma(\beta)/\partial\beta = 0$ we find that the optimal $\beta$ solves

$$\sigma_\beta^2 \equiv \sigma^2\left(\frac{1}{\beta N} + 1\right) = \sigma^2\left(\frac{1}{N} + 1\right) \quad \Rightarrow \quad \beta = 1 \qquad (16)$$

Note that this result holds for any $N$ and is independent of the teacher parameter. The Bayes averaging at unit temperature is optimal for any given value of $\theta_0$, hence, for any teacher distribution. We may say that the vague Bayes scheme is robust to the teacher distribution in this case. Clearly this is a much stronger optimality than the more general result proven above.

## 3.2   Bias-variance tradeoff

It is interesting to decompose the generalization error in Eq. 15 in bias and variance components. We follow Heskes [7] and define the bias error as the generalization error of the geometric average distribution,

$$B(\beta) \equiv \int -\log\overline{p}(x)p(x|\theta_0)dx, \qquad (17)$$

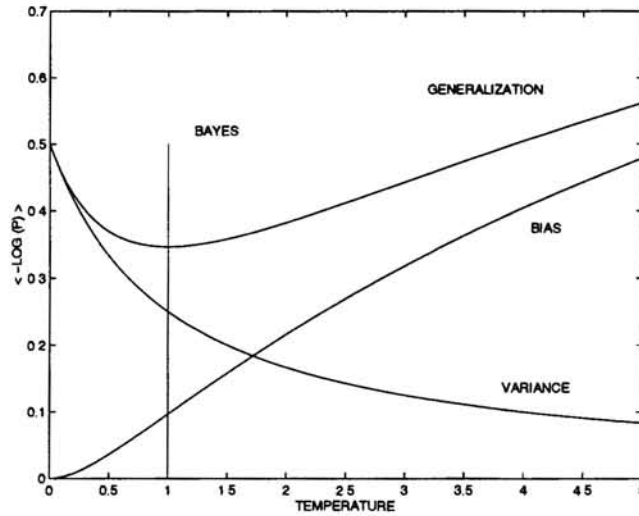

Figure 1: Bias-variance trade-off as function of the width of the temperated likelihood ensemble (temperature $= 1/\beta$) for $N = 1$. The bias is computed as the generalization error of the predictive distribution obtained from the geometric average distribution w.r.t. training set fluctuations as proposed by Heskes. The predictive distribution produced by Bayesian averaging corresponds to unit temperature (vertical line) and it achieves the minimal generalization error. Maximum-likelihood estimation for reference is recovered as the zero width/temperature limit.

with

$$\bar{p}(x) = Z^{-1} \exp \left( \int \log[p(x|D)] p(D|\theta_0) dD \right). \tag{18}$$

Inserting from Eq. (13), we find

$$\bar{p}(x) = \frac{1}{\sqrt{2\pi\sigma_\beta^2}} \exp \left( -\frac{1}{2\sigma_\beta^2} (x - \theta_0)^2 \right). \tag{19}$$

Integrating over the teacher distribution we find,

$$B(\beta) = \frac{1}{2} \log 2\pi\sigma_\beta^2 + \frac{\sigma^2}{2\sigma_\beta^2} \tag{20}$$

The variance error is given by $V(\beta) = \Gamma(\beta) - B(\beta)$,

$$V(\beta) = \frac{\sigma^2}{2N\sigma_\beta^2} \tag{21}$$

We can now quantify the statements above. By averaging a bias is introduced –the predictive distribution becomes wider– which decrease the variance contribution initially so that the generalization error being the sum of the two decreases. At still higher temperatures the bias becomes too strong and the generalization error start to increase. The Bayes average at unit temperature is the optimal trade-off within the given family of procedures.

### 3.3   Asymptotics for smoothly parameterized models

We now go on to show that a similar result also holds for general learning problems in limit of large data sets. We consider a system parameterized by a finite dimensional parameter vector $\theta$. For a given large training set and for a smooth likelihood function, the tempered likelihood is approximately Gaussian centered at the maximum posterior parameters[13], hence the normalized tempered posterior reads

$$P(\theta|\beta D, H) = \sqrt{\left|\frac{\beta N A(D, \theta_{ML})}{2\pi}\right|} \exp\left(-\frac{\beta N}{2}\delta\theta' A(D, \theta_{ML})\delta\theta\right) \qquad (22)$$

where $\delta\theta = \theta - \theta_{ML}$, with $\theta_{ML} = \theta_{ML}(D)$ denoting the maximum likelihood solution for the given training sample. The second derivative or *Hessian* matrix is given by

$$A(D, \theta) = \frac{1}{N}\sum_{\alpha=1}^{N} A(x_\alpha, \theta) \qquad (23)$$

$$A(x, \theta) = \frac{\partial^2}{\partial\theta\partial\theta'} - \log p(x|\theta) \qquad (24)$$

The predictive distribution is given by

$$p(x|\beta, D) = \int p(x|\theta)p(\theta|\beta, D)d\theta \qquad (25)$$

we write $p(x|\theta) = \exp(-\epsilon(x|\theta))$ and expand $\epsilon(x|\theta)$ around $\theta_{ML}$ to second order, we find

$$p(x|\theta) \approx p(x|\theta_{ML}) \exp\left(-a(x|\theta_{ML})'\delta\theta - \tfrac{1}{2}\delta\theta' A(x|\theta_{ML})\delta\theta\right). \qquad (26)$$

We are then in position to perform the integration over the posterior to find the normalized predictive distribution,

$$p(x|\beta, D) = p(x|\theta_{ML})\sqrt{\frac{|\beta N A(D)|}{|\beta N A(D) + A(x)|}} \exp\left(\tfrac{1}{2}a(x|\theta_{ML})' A(x|\theta_{ML})a(x|\theta_{ML})\right). \qquad (27)$$

Proceeding as above, we compute the generalization error

$$\Gamma(\beta, \theta_0) = \int\int -\log p(x|\beta, D)p(x|\theta_0)dx\, p(D|\theta_0)dD \qquad (28)$$

For sufficiently smooth likelihoods, fluctuations in the maximum likelihood parameters will be asymptotic normal, see e.g. [8], and furthermore fluctuations in $A(D)$ can be neglected, this means that we can approximate,

$$A(x) + A(D) \approx (\frac{1}{N} + 1)A_0, \quad A_0 = \int A(x|\theta_0)p(x|\theta_0)dx \qquad (29)$$

where $A_0$ is the averaged Fisher information matrix. With these approximations (valid as $N \to \infty$) the generalization error can be found,

$$\Gamma(\beta, \theta_0) \approx \Gamma(\infty) + \frac{d}{2}\log\left(1 + \frac{1}{\beta N}\right) - \frac{d}{2}\frac{1 + \frac{1}{N}}{1 + \beta N}. \qquad (30)$$

with $d = \dim(\theta)$ denoting the dimension of the parameter vector. Like in the 1D example (Eq. (15)) we find the generalization error is asymptotically independent of the teacher parameters. It is minimized for $\beta = 1$ and we conclude that Bayes is well-tempered in the asymptotics and that this holds for any teacher distribution. In the Bayes literature this is refered to as the prior is overwhelmed by data [1]. Decomposing the errors in bias and variance contributions we find similar results as for in 1D example, Bayes introduces the optimal bias by averaging at unit temperature.

# 4 Discussion

We have seen two examples of Bayes averaging being optimal, in particular improving on maximum likelihood estimation. We found that averaging introduces a bias and reduces variance so that the generalization error (being the sum of bias and variance) initially decrease. Bayesian averaging at unit temperature is the optimal width of the averaging distribution. For larger temperatures (widths) the bias is too strong and the generalization error increases. Both examples were special in the sense that they lead to generalization errors that are independent of the random teacher parameter. This is not generic, of course, rather the generic case is that a mis-specified prior can lead to arbitrary large learning catastrophes.

## Acknowledgments

I thank the organizers of the 1999 Max Planck Institute Workshop on Statistical Physics of Neural Networks Michael Biehl, Wolfgang Kinzel and Ido Kanter, where this work was initiated. I thank Carl Edward Rasmussen, Jan Larsen, and Manfred Opper for stimulating discussions on Bayesian averaging. This work was funded by the Danish Research Councils through the Computational Neural Network Center CONNECT and the THOR Center for Neuroinformatics.

## Footnotes

[1]This does not limit us to conventional density estimation; pattern recognition and many functional approximations problems can be formulated as density estimation problems as well.

# References

[1] C.P. Robert: *The Bayesian Choice - A Decision-Theoretic Motivation.* Springer Texts in Statistics, Springer Verlag, New York (1994). A. Ohagan: *Bayesian Inference.* Kendall's Advanced Theory of Statistics. Vol 2B. The University Press, Cambridge (1994).

[2] L. Breiman: *Using adaptive bagging to debias regressions.* Technical Report 547, Statistics Dept. U.C. Berkeley, (1999).

[3] R.T. Clemen *Combining forecast: A review and annotated bibliography.* Journal of Forecasting **5**, 559 (1989).

[4] L.K. Hansen and P. Salamon: *Neural Network Ensembles.* IEEE Transactions on Pattern Analysis and Machine Intelligence, **12**, 993-1001 (1990).

[5] L.K. Hansen: *Stochastic Linear Learning: Exact Test and Training Error Averages.* Neural Networks 6, 393-396, (1993)

[6] D. Haussler and M. Opper: *Mutual Information, Metric Entropy, and Cumulative Relative Entropy Risk* Annals of Statistics **25** 2451-2492 (1997)

[7] T. Heskes: *Bias/Variance Decomposition for Likelihood-Based Estimators.* Neural Computation **10**, pp 1425-1433, (1998).

[8] L. Ljung: *System Identification: Theory for the User.* Englewood Cliffs, New Jersey: Prentice-Hall, (1987).

[9] J. Moody: "Note on Generalization, Regularization, and Architecture Selection in Nonlinear Learning Systems," in B.H. Juang, S.Y. Kung & C.A. Kamm (eds.) *Proceedings of the first IEEE Workshop on Neural Networks for Signal Processing*, Piscataway, New Jersey: IEEE, 1–10, (1991).

[10] N. Murata, S. Yoshizawa & S. Amari: *Network Information Criterion — Determining the Number of Hidden Units for an Artificial Neural Network Model.* IEEE Transactions on Neural Networks, vol. 5, no. 6, pp. 865–872, 1994.

[11] V. Vapnik: *Estimation of Dependences Based on Empirical Data.* Springer-Verlag New York (1982).

[12] H. White, "Consequences and Detection of Misspecified Nonlinear Regression Models," *Journal of the American Statistical Association*, **76**(374), 419–433, (1981).

[13] D.J.C MacKay: *Bayesian Interpolation*, Neural Computation **4**, 415-447, (1992).